# Discriminative Densities from Maximum Contrast Estimation

**Peter Meinicke**
Neuroinformatics Group
University of Bielefeld
Bielefeld, Germany
*pmeinick@techfak.uni-bielefeld.de*

**Thorsten Twellmann**
Neuroinformatics Group
University of Bielefeld
Bielefeld, Germany
*ttwellma@techfak.uni-bielefeld.de*

**Helge Ritter**
Neuroinformatics Group
University of Bielefeld
Bielefeld, Germany
*helge@techfak.uni-bielefeld.de*

## Abstract

We propose a framework for classifier design based on *discriminative densities* for representation of the *differences* of the class-conditional distributions in a way that is optimal for classification. The densities are selected from a parametrized set by constrained maximization of some objective function which measures the average (bounded) difference, i.e. the *contrast* between discriminative densities. We show that maximization of the contrast is equivalent to minimization of an approximation of the Bayes risk. Therefore using suitable classes of probability density functions, the resulting *maximum contrast classifiers* (MCCs) can approximate the Bayes rule for the general multiclass case. In particular for a certain parametrization of the density functions we obtain MCCs which have the same functional form as the well-known Support Vector Machines (SVMs). We show that MCC-training in general requires some nonlinear optimization but under certain conditions the problem is concave and can be tackled by a single linear program. We indicate the close relation between SVM- and MCC-training and in particular we show that Linear Programming Machines can be viewed as an approximate realization of MCCs. In the experiments on benchmark data sets, the MCC shows a competitive classification performance.

## 1 Introduction

In the Bayesian framework of classification the ultimate goal of a classifier $f(\mathbf{x}) : \mathbb{R}^D \to \{1, \dots, M\}$ is to minimize the expected risk of misclassification measured by $l(i, j)$ which denotes the loss for assigning a given feature vector to class $j$, while it actually belongs to class $i$, with $M$ being the number of classes. With $p(\mathbf{x} \mid i)$ being the class-conditional probability density functions (PDFs) and $\pi_i$ denoting the corresponding apriori probabilities of

class-membership we have the risk

$$\mathcal{R} = E[l(y, f(\mathbf{x}))] = \sum_{i=1}^{M} \pi_i \int l(i, f(\mathbf{x})) p(\mathbf{x} \mid i) \, d\mathbf{x}. \tag{1}$$

With the standard "zero-one" loss function $l(i, j) = 1 - \delta_{ij}$, where $\delta_{ij}$ denotes the Kronecker delta, it is easy to show (see e.g. [3]) that the expected risk is minimized, if one chooses the classifier

$$f(\mathbf{x}) = \arg \max_i \pi_i p(\mathbf{x} \mid i) \tag{2}$$

The resulting lower bound on $\mathcal{R}$ is known as the *Bayes risk* which limits the average performance of the classifier $f(\mathbf{x})$. Because the class-conditional densities are usually unknown, one way to realize the above classifier is to use estimates of these densities instead. This leads to the so-called *plug-in* classifiers, which are Bayes-consistent if the density estimators are consistent (e.g. [9]). Due to the notoriously slow convergence of density estimates the plug-in scheme usually isn't the best recipe for classifier design and as an alternative many discriminant functions including Neural Networks (see [1, 9] for an overview) and Support Vector Machines (SVMs) [2, 12] have been proposed which are trained directly to minimize the empirical classification error.

We recently proposed a method for the design of density-based classifiers without resorting to the usual density estimation schemes of the plug-in approach [6]. Instead we utilized *discriminative densities* with parameters optimized to solve the classification problem. The approach requires maximization of the average bounded difference between class (discriminative) densities $p(\mathbf{x}_i \mid \theta_j)$, which we refer to as the *contrast* of the underlying "true" distributions. The $\gamma_{\max}$-bounded contrast is the expectation $E[\mathcal{C}(\mathbf{x}, y; \gamma_{\max})]$ with

$$\widetilde{\mathcal{C}}(\mathbf{x}, y; \gamma_{\max}) = \sum_{j \neq y} \min\{\gamma_{\max}, \ \pi_y p(\mathbf{x}; \boldsymbol{\theta}_y) - \pi_j p(\mathbf{x}; \boldsymbol{\theta}_j)\}. \tag{3}$$

The idea is to find $M$ discriminative densities $p(\mathbf{x}; \boldsymbol{\theta}_i)$, which represent the underlying distributions with "true" densities $p(\mathbf{x} \mid i)$ in a way, that is optimal for classification. When maximizing the contrast with respect to the parameters $\boldsymbol{\theta}_i$ of the discriminative densities the upper bound $\gamma_{\max}$ plays a central role because it prevents the learning algorithm from increasing the differences between discriminative densities where the differences between the true densities are already large.

In this paper we show that with some slight modification the contrast can be viewed as an approximation of the negative Bayes risk (up to some constant shift and scaling) which is valid for the binary as well as for the general multiclass case. Therefore for certain parametrizations of the discriminative densities MCCs allow to find an optimal trade-off between the classical plug-in Bayes-consistency and the consistency which arises from direct minimization of the approximate Bayes risk. Furthermore, for a particular parametrization of the PDFs, we obtain certain kinds of Linear Programming Machines (LPMs) [4] as (in general) approximate solutions of maximum contrast estimation. In that way MCCs provide a Bayes-consistent approach to realize multiclass LPMs / SVMs and they suggest an interpretation of the magnitude of the LPM / SVM classification function in terms of density differences which provide a probabilistic measure of confidence. For the case of LPMs we propose an extended optimization procedure for maximization of the contrast via iteration of linear optimizations. Inspired by the MCC-framework, for the resulting *Sequential Linear Programming Machines* (SLPM) we propose a new regularizer which allows to find an optimal trade-off between the above mentioned two approaches to Bayes consistency. In the experiments we analyse the performance of the SLPM on simulated and real world data.

## 2 Maximum Contrast Estimation

For the design of MCCs the first step, which is the same as for the plug-in concept, requires to replace the unknown class-conditional densities of the Bayes classifier (2) by suitably parametrized PDFs. Then, instead of choosing the parameters for an approximation of the original (true) densities (e.g. by maximum likelihood estimation) as with the plug-in scheme, the density parameters are choosen to maximize the so-called *contrast* which is the expected value of the $\gamma_{\max}$-bounded density differences as defined in (3).

For the case of an unbounded contrast, i.e. $\gamma_{\max} \rightarrow \infty$, the general maximum contrast solution can be found analytically and for notational simplicity we derive it for the binary case with equal apriori probabilities, where the contrast can be written as

$$\int (p_1(\mathbf{x}) - p_2(\mathbf{x}))p(\mathbf{x} \mid 1)\, d\mathbf{x} + \int (p_2(\mathbf{x}) - p_1(\mathbf{x}))p(\mathbf{x} \mid 2)\, d\mathbf{x}$$

$$= \int (p(\mathbf{x} \mid 1) - p(\mathbf{x} \mid 2))p_1(\mathbf{x})\, d\mathbf{x} + \int (p(\mathbf{x} \mid 2) - p(\mathbf{x} \mid 1))p_2(\mathbf{x})\, d\mathbf{x}.$$

Thus the unbounded contrast is maximized for $\hat{p}_1(\mathbf{x}) = \delta(\mathbf{x} - \mathbf{x}_1), \hat{p}_2(\mathbf{x}) = \delta(\mathbf{x} - \mathbf{x}_2)$ with the peaks of the Delta (Dirac) functions located at $\mathbf{x}_1 = \arg\max_{\mathbf{x}}(p(\mathbf{x} \mid 1) - p(\mathbf{x} \mid 2))$ and $\mathbf{x}_2 = \arg\max_{\mathbf{x}}(p(\mathbf{x} \mid 2) - p(\mathbf{x} \mid 1))$, respectively. Obviously, these are not the best discriminative densities we may think of and therefore we require an appropriate bound $\gamma_{\max}$. For finite $\gamma_{\max}$, maximization of the contrast enforces a redistribution of the estimated probability mass and gives rise to a constrained linear optimization problem in the space of discriminative densities which may be solved by variational methods in some cases.

The relation between contrast and Bayes risk becomes more convenient when we slightly modify the above definition (3) by a unit upper bound and by adding a *lower* bound on the $\eta$-scaled density differences:

$$\mathcal{C}(\mathbf{x}, y; \eta) = \max\{-1, \frac{1}{M-1}\sum_{j \neq y} \min\{1, \eta \cdot [\pi_y p(\mathbf{x}; \boldsymbol{\theta}_y) - \pi_j p(\mathbf{x}; \boldsymbol{\theta}_j)]\}\} \quad (4)$$

with scale factor $\eta = 1/\gamma_{\max}$. Therefore, for an infinite scale factor $\eta$ the (expected) contrast $\mathcal{C}(\eta) = E[\mathcal{C}(\mathbf{x}, y; \eta)]$ approaches the negative Bayes risk up to constant shift and scaling:

$$\mathcal{R} = \lim_{\eta \to \infty} E[\frac{1}{2} - \frac{1}{2}\mathcal{C}(\mathbf{x}, y; \eta)] = \frac{1}{2} - \frac{1}{2}\lim_{\eta \to \infty} \mathcal{C}(\eta). \quad (5)$$

Thus the scale factor defines a subset of the input-space, which includes the decision boundary and which becomes increasingly focused in their vicinity as $\eta \to \infty$. The extent of the region is defined by the bounds $\pm 1/\eta$ on the difference between discriminative densities. In terms of the contrast function it can be defined as

$$\mathcal{S} = \{\mathbf{x} : \exists y : |\mathcal{C}(\mathbf{x}, y; \eta)| < 1\}. \quad (6)$$

Since for MCC-training we maximize the *empirical* contrast, i.e. the corresponding sample average of $\mathcal{C}(\mathbf{x}, y; \eta)$, the scale factor then defines a subset of the training data which has impact on learning of the decision boundary. Thus for increasing scale factor the relative size of that subset is shrinking. However for increasing size of the training set the scale factor can be gradually increased and then, for suitable classes of PDFs, MCCs can approach the Bayes rule. In other words, $\eta$ acts as a regularization parameter such that, for particular choices of the PDF class convergence to the Bayes classifier can be achieved if the quality of the approximation of the loss function is gradually increased for increasing sample sizes. In the following section we shall consider such a class of PDFs which is flexible enough and which turns out to include a certain kind of SVMs.

# 3 MCC-Realizations

In the following we shall first consider a particularly useful parametrization of the discriminative densities which gives rise to classifiers which in the binary case have the same functional form as SVMs up to a "missing" bias term in the MCC-case. For training of these MCCs we derive a suitable objective function which can be maximized by sequential linear programming where we show the close relation to training of Linear Programming Machines.

## 3.1 Density Parametrization

We first have to choose a set of candidate functions from which we select the required PDF. Because this set should provide some flexibility with respect to contrast maximization the usual kernel density estimator (KDE)[11]

$$\hat{p}(\mathbf{x} \mid j) = \frac{1}{|I_j|} \sum_{i \in I_j} K(\mathbf{x}, \mathbf{x}_i) \tag{7}$$

with index set $I_j$ containing indices of examples from class $j$ and with normalized kernel functions according to $\int K(\mathbf{x}, \cdot) d\mathbf{x} = 1$ isn't a quite good choice, since the only free parameter is the kernel bandwidth which doesn't allow for any *local* adaptation. On the other hand if we allow for local variation of the bandwidth we get a complicated contrast which is difficult to maximize due to nonlinear dependencies on the parameters. The same is true if we treat the kernel centers as free parameters. However, if we modify the kernel density estimator to have flexible mixing weights according to

$$p(\mathbf{x}; \boldsymbol{\theta}_j) = \sum_{i \in I_j} \omega_{ij} K(\mathbf{x}, \mathbf{x}_i) = \boldsymbol{\omega}_j^T \mathbf{k}_j(\mathbf{x}) \quad \text{with } \|\boldsymbol{\omega}_j\|_1 = 1, \ \boldsymbol{\omega}_j \geq \mathbf{0} \tag{8}$$

we get an objective function, which is linear in the mixing parameters $\omega_{ij}$ under certain conditions. Thus we have class-specific densities with mixing weights $\omega_{ij}$ which control the contribution of a single training example to the PDF.

With that choice we achieve plug-in Bayes-consistency for the case of equal mixing weights, since then we have the usual kernel density estimator (KDE), which, besides some mild assumptions about the distributions, requires a vanishing kernel bandwidth for $N \to \infty$.

## 3.2 Objective Function

For notational simplicity in the following we shall incorporate the scale factor $\eta$ and the mixing weigths $\boldsymbol{\omega}_j$ into a common parameter vector $\boldsymbol{\alpha} = (\boldsymbol{\alpha}_1^T, \ldots, \boldsymbol{\alpha}_M^T)^T$ with $\boldsymbol{\alpha}_j = \eta \boldsymbol{\omega}_j$ and $\|\boldsymbol{\alpha}_j\|_1 = \|\boldsymbol{\alpha}_k\|_1 > 0 \forall j, k$. Further we define the scaled density difference

$$D_{ij}(\mathbf{x}; \boldsymbol{\alpha}) = \pi_i \boldsymbol{\alpha}_i^T \mathbf{k}_i(\mathbf{x}) - \pi_j \boldsymbol{\alpha}_j^T \mathbf{k}_j(\mathbf{x}). \tag{9}$$

so that we can write the empirical contrast $\mathcal{C}_N(\boldsymbol{\alpha})$, i.e. the sample average over $N$ training examples, as:

$$\mathcal{C}_N(\boldsymbol{\alpha}) = \frac{1}{N} \sum_{j=1}^{M} \sum_{i \in I_j} \left[ \beta_i \left( 1 + \frac{1}{M-1} \sum_{k \neq j} \min\{1, D_{jk}(\mathbf{x}_i; \boldsymbol{\alpha})\} \right) - 1 \right] \tag{10}$$

where the assignment variables $\beta_i \in \{0, 1\}$ realize the maximum function in (4). With fixed assignment variables $\beta_i$, $\mathcal{C}_N$ is concave and maximization with respect to $\boldsymbol{\alpha}$ gives rise

to a linear optimization problem. On the other hand, for fixed $\boldsymbol{\alpha}$ maximization with respect to the $\beta_i$ is achieved by setting $\beta_i = 0$ for negative terms. This suggests a sequential linear optimization strategy for overall maximization of the contrast which shall be introduced in detail in the following section.

Since we have already incorporated $\eta$ as a scaling factor into the parameter vector $\boldsymbol{\alpha}$, $\eta$ is now identified with the norm $\|\boldsymbol{\alpha}_j\|_1$. Therefore the scale factor can be adjusted implicitly by a regularization term which penalizes some suitable norm of the $\boldsymbol{\alpha}_j$. Thus a suitable objective function can be defined by

$$\mathcal{C}_N(\boldsymbol{\alpha}, \lambda) = \mathcal{C}_N(\boldsymbol{\alpha}) - \lambda \mathcal{P}(\boldsymbol{\alpha}), \quad \lambda > 0 \tag{11}$$

with $\lambda$ determining the weight of the penalty, i.e. the degree of regularization. We now consider several instances of the case where the penalty corresponds to some $p$-norm of $\boldsymbol{\alpha}$. With the 1-norm, for $\lambda \to \infty$ the probability mass of the discriminative densities is concentrated on those two kernel-functions which yield the highest average density difference. Although that property forces the sparsest solution for large enough $\lambda$, clearly, that solution isn't Bayes-consistent in general because as pointed out in Sec.2, for $\lambda \to \infty$ all probability mass of the discriminative densities is concentrated at the two points with maximum average density difference.

Conversely taking $\mathcal{P}(\boldsymbol{\alpha}) = \|\boldsymbol{\alpha}\|_2^2$, which resembles the standard SVM regularizer [10], yields the KDE with equal mixing weights for $\lambda \to \infty$. Indeed, it is easy to see that all $p$-norm penalties with $p > 1$ share this convenient property, which guarantees "plug-in" Bayes consistency in the case where the solution is totally determined by the regularizer. In that case kernel density estimators are achieved as the "default" solution. Therefore we chose a combination of the 1-norm with the maximum-norm

$$\mathcal{P}(\boldsymbol{\alpha}) = \sum_{j=1}^{M} (\|\boldsymbol{\alpha}_j\|_1 + \|\boldsymbol{\alpha}_j\|_\infty) \tag{12}$$

which is easily incorporated into a linear program, as to be shown in the following. For that kind of penalty in the limiting case $\lambda \to \infty$ we achieve an equal distribution of the weights which corresponds to the kernel density estimator (KDE) solution. In that way we have a nice trade-off between two kinds of Bayes consistency: for increasing $\lambda$ the class-specific densities converge to the KDE with equal mixing weights, whereas for decreasing $\lambda$ the probability mass of the discriminative densities is more and more concentrated near the Bayes-optimal decision boundary. By a suitable choice of the kernel width and the scale of the weights, e.g. via cross-validation, the solution with fastest convergence to the Bayes rule may be selected.

With an 1-norm penalty on the weights and on the vector $\boldsymbol{\xi}$ of soft margin slack variables we get the Linear Programming Machine which requires to minimize

$$\|\boldsymbol{\alpha}\|_1 + C\|\boldsymbol{\xi}\|_1 \quad \text{subject to} \quad y_i \sum_{j=1}^{N} y_j \alpha_j K(\mathbf{x}_i, \mathbf{x}_j) \geq 1 - \xi_i, \ \xi_i \geq 0 \tag{13}$$

with $y \in \{-1, 1\}$ and with the above constraints on $\boldsymbol{\alpha}$. Dividing the objective by $C$, subtracting $N$, setting $1 - \xi_i \equiv \gamma_i$ and turning minimization to maximization of the negative objective shows that LPM training corresponds to a special case of MCC training with fixed $\beta_i = 1$ and 1-norm regularizer with $\lambda = 1/C$.

## 3.3 Sequential Linear Programming

Estimation of mixing weights is now achieved by maximizing the sample contrast with respect to the $\alpha_{ij}$ and the assignment variables $\beta_i$. This can be achieved by the following iterative optimization scheme:

1. **Initialization**: $\beta_i = 1 \ \forall i$

2. **Maximization w.r.t. $\boldsymbol{\alpha}$ for fixed $\boldsymbol{\beta}$**:

$$\text{maximize} \sum_{j=1}^{M} \sum_{i \in I_j} \beta_i \sum_{k \neq j} \gamma_{ijk} - \lambda \sum_{j=1}^{M} (\alpha_{\max}^{(j)} + \sum_{i \in I_j} \alpha_{ij})$$

$$\text{subject to } \gamma_{ijk} \leq 1, \quad D_{jk}(\mathbf{x}_i, \boldsymbol{\alpha}) - \gamma_{ijk} \geq 0, \ k \neq j,$$

$$\|\boldsymbol{\alpha}_j\|_1 = \|\boldsymbol{\alpha}_k\|_1 \geq 1 \ \forall j, k \quad \alpha_{\max}^{(j)} \geq \alpha_{ij} \geq 0 \ \forall i, j$$

3. **Maximization w.r.t. $\boldsymbol{\beta}$ for fixed $\boldsymbol{\alpha}$**:

$$\beta_i = \begin{cases} 1, & \sum_{j \neq y_i} D_{jk}(\mathbf{x}_i, \boldsymbol{\alpha}) > 1 - M \\ 0, & \text{otherwise.} \end{cases}$$

4. **If** convergence in contrast **then** stop **else** proceed with step 2.

Where $\gamma_{ijk}$ are slack variables, measuring the part of the density difference $D_{jk}(\mathbf{x}_i, \boldsymbol{\alpha})$ which can be charged to the objective function. The constraint $\|\boldsymbol{\alpha}_j\|_1 \geq 1$ in the linear program was chosen in order to prevent the trivial solution $\boldsymbol{\alpha} = \mathbf{0}$ which may otherwise appear for larger values of $\lambda$. Since we used unnormalized Gaussian kernel functions with $K(\mathbf{x}, \mathbf{x}) = 1$, i.e. we excluded all multiplicative density constants, that constraint doesn't exclude any useful solutions for the weights.

## 4 Experiments

In the following section we consider the task of solving binary classification problems within the MCC-framework, using the above SLPM with Gaussian kernel function. The first experiment illustrates the behaviour of the MCC for different values for the regularization $\lambda$ by means of a simple two-dimensional toy dataset. The second experiment compares the classification performance of the MCC with those of the SVM and *Kernel-Density-Classifier* (*KDC*) which is a special case of the MCC with equal weighting of each kernel function. To this end, we selected four frequently used benchmark datasets from the UCI Machine Learning Repository.

The two-dimensional toy dataset consists of 300 data points, sampled from two overlapping isotropic normal distributions with a mutual distance of $d = 2.8$ and standard deviation $\frac{d}{3}$. Figure 1 shows the solution of the MCC for two different values of $\lambda$ (only data points with non-zero weights according the criterion $\alpha_i > 10^{-6}$ are marked by symbols). In both figures, data points with large mixing weights are located near the decision border. In particular for small $\lambda$ there are regions of high contrast $|\mathcal{C}|$ alongside the decision function (illustrated by isolines). For increasing $\lambda$ the number of data points with non-zero $\alpha_i$ increases. At the same time, one can note a decrease of the difference between the weights. Regions with contrast $|\mathcal{C}| \geq 1$ are highlighted gray. For small values of $\lambda$, these regions are nearer to the decision border than for large values. This illustrates that for increasing $\lambda$ the quality of the approximation of the loss function decreases. In both figures, several data points are misclassified with a contrast $|\mathcal{C}| \geq 1$. The MCC identified those data points as outliers and deactivated them during the training (encircled symbols).

The second experiment demonstrates the performance of the MCC in comparison with those of a Support Vector Machine, as one of the state-of-the-art binary classifiers, and with the KDC. For this experiment we selected the *Pima Indian Diabetes*, *Breast-Cancer*, *Heart* and *Thyroid* dataset from the UCI Machine Learning repository. The Support Vector Machine was trained using the *Sequential Minimal Optimization* algorithm by J. Platt[7] adjusted according to the modification proposed by S.S. Keerthi [5].

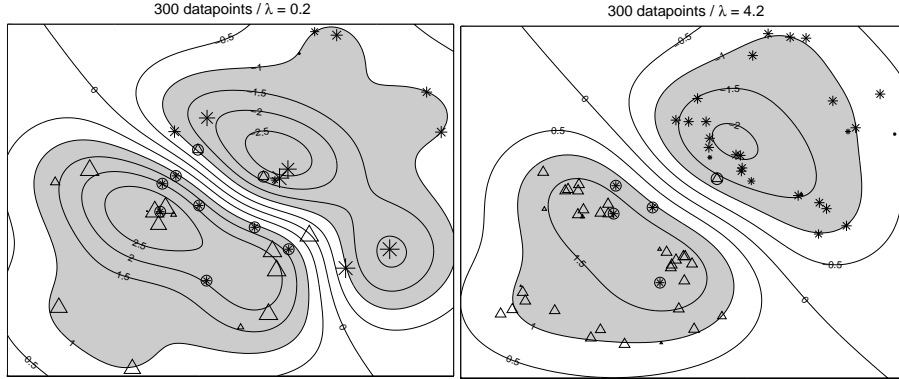

Figure 1: Two MCC solutions for the two-dimensional toy dataset for different values of $\lambda$ (left: $\lambda = 0.2$, right: $\lambda = 4.2$). The symbols $*$ and $\triangle$ depict the positions of data points with with non-zero $\alpha_i$. The size of each symbol is scaled according the value of the corresponding $\alpha_i$. Encircled symbols have been deactivated during the training (symbols for deactivated data points are not scaled according to $\alpha_i$, since in most cases $\alpha_i$ is zero). The absolute value of the contrast is illustrated by the isolines while the sign of the contrast depicts the binary classification of the classifier. The region with $|\mathcal{C}| < 1$ which corresponds to $\mathcal{S}$ as defined in (6) is colored white and the complement colored gray. The percentage of data points that define the solution is $8.7\%$ (left figure) and $20\%$ (right figure) of the dataset.

The experimental setup was comparable with that in [8]: After normalization to zero mean and unit standard deviation, each dataset was divided 100 times in different pairs of disjoint train- and testsets with a ratio of $60\%{:}40\%$ (provided by G. Rätsch at http://ida.first.gmd.de/~raetsch/data/benchmarks.htm). Since we used for all classifiers the Gaussian kernel function, all three algorithms are parametrized by the bandwidth $\sigma$. Additionally, for the SVM and MCC the regularization value $\lambda$ had to be chosen. The optimal parametrization was chosen by estimating the generalization performance for different values of bandwidth and regularization by means of the average test error on the first five dataset partitions. More precisely, a first coarse scan was performed, followed by a fine scan in the interval near the optimal values of the first one. Each scan considered 1600 different combinations of $\sigma$ and $\lambda$, resp. $\sigma$ and $C$. For parameter pairs with identical test error, the pair constructing the sparsest solution was kept. Finally, the reported values in Tab.1 and Tab.2 are averaged over all 100 dataset partitions.

Table 1 shows the optimal parametrization $(\sigma, \lambda)$ of the MCC in combination with the classification rate and sparseness of the solution (measured as percentage non-zero $\alpha_i$). Additionally, the corresponding values after the first MCC iteration are given in brackets. The last two columns show the absolute number of iterations and the final number of de-activated examples. For all four datasets the MCC is able to find a sparse solution. In particular for the Heart, Breast-Cancer and Diabetes dataset the solution of the MCC is significantly sparser than those of the SVM (see Tab.2). Nevertheless, Tab.2 indicates that the classification rates of the MCC are competitive with those of the SVM.

## 5  Conclusion

The MCC-approach provides an understanding of SVMs / LPMs in terms of generative modelling using *discriminative densities*. While usual unsupervised density estimation schemes try to minimize some distance criterion (e.g. Kullback-Leibler divergence) be-

Table 1: Optimal parametrization $(\sigma, \lambda)$, classification rate, percentage of non-zero $\alpha_i$, number of iterations of the MCC and number of $\beta_i = 0$. The results are averaged over all 100 dataset partitions. For the classification rate and percentage of non-zero $\alpha$-coefficients the corresponding value after the first MCC iteration is given in brackets.

| Dataset | $\sigma$ | $\lambda$ | Classif. rate | $\#\alpha > 10^{-6}$ | #Iter. | $\#\beta = 0$ |
|---|---|---|---|---|---|---|
| Breast-Cancer | 1.38 | 12.17 | 74.3% (74.4%) | 13.6% (13.8%) | 2.23 | 2.6 |
| Heart | 2.69 | 2.066 | 84.3% (84.1%) | 20.4% (21.2%) | 3.10 | 6.4 |
| Thyroid | 0.49 | $10^{-5}$ | 95.5% (95.5%) | 46.1% (46.1%) | 1.00 | 0.0 |
| Diabetes | 4.52 | 2.624 | 76.6% (76.5%) | 5.3% (5.5%) | 5.86 | 40.7 |

Table 2: Summary of the performance of the KDC, SVM and MCC for the four benchmark datasets. Given are the classification rates with percentage of non-zero $\alpha_i$ (in brackets). Note that our results for the SVM are slightly better to those reported in [8]. One reason could be the coarse parameter selection for the SVM as already mentioned by the author.

| Dataset | KDC | SVM | MCC |
|---|---|---|---|
| Breast-Cancer | 73.1 % (100 %) | 74.5 % (58.5 %) | 74.3 % (13.6 %) |
| Heart | 84.1 % (100 %) | 84.4 % (60.9 %) | 84.3 % (20.4 %) |
| Thyroid | 95.6 % (100 %) | 95.7 % (15.8 %) | 95.5 % (46.1 %) |
| Diabetes | 74.2 % (100 %) | 76.7 % (53.6 %) | 76.6 % ( 5.3 %) |

tween the models and the true densities, MC-estimation aims at learning of densities which represent the differences of the underlying distributions in an optimal way for classification. Future work will address the investigation of the general multiclass performance and the capability to cope with misslabeled data.

# References

[1] C. M. Bishop. *Neural Networks for Pattern Recognition*. Clarendon Press, Oxford, 1995.

[2] C. Cortes and V. Vapnik. Support-vector networks. *Machine Learning*, 20(3):273–297, 1995.

[3] R. O. Duda and P. E. Hart. *Pattern Classification and Scene Analysis*. Wiley, New York, 1973.

[4] T. Graepel, R. Herbrich, B. Scholkopf, A. Smola, P. Bartlett, K. Robert-Muller, K. Obermayer, and B. Williamson. Classification on proximity data with lp–machines, 1999.

[5] S.S. Keerthi, S.K. Shevade, C. Bhattacharyya, and K.R.K. Murthy. Improvements to platt's SMO algorithm for SVM classifier design. Technical report, Dept of CSA, IISc, Bangalore, India, 1999.

[6] P. Meinicke, T. Twellmann, and H. Ritter. Maximum contrast classifiers. In *Proc. of the Int. Conf. on Artificial Neural Networks*, Berlin, 2002. Springer. in press.

[7] J. Platt. Fast training of support vector machines using sequential minimal optimization. In B. Schölkopf, C. J. C. Burges, and A. J. Smola, editors, *Advances in Kernel Methods — Support Vector Learning*, pages 185–208, Cambridge, MA, 1999. MIT Press.

[8] G. Rätsch, T. Onoda, and K.-R. Müller. Soft margins for AdaBoost. Technical Report NC-TR-1998-021, Department of Computer Science, Royal Holloway, University of London, Egham, UK, August 1998. Submitted to Machine Learning.

[9] B. D. Ripley. *Pattern Recognition and Neural Networks*. Cambridge University Press, Cambridge, 1996.

[10] B. Schölkopf and A. J. Smola. *Learning with Kernels*. MIT Press, 2002.

[11] D. W. Scott. *Multivariate Density Estimation*. Wiley, 1992.

[12] V. N. Vapnik. *The Nature of Statistical Learning Theory*. Springer, New York, 1995.
